# Variational Learning for Recurrent Spiking Networks

**Danilo Jimenez Rezende**
Brain Mind Institute
École Polytechnique Fédérale de Lausanne
1015 Lausanne EPFL, Switzerland
`danilo.rezende@epfl.ch`

**Daan Wierstra**
School of Computer and Communication Sciences, Brain Mind Institute
École Polytechnique Fédérale de Lausanne
1015 Lausanne EPFL, Switzerland
`daan.wierstra@epfl.ch`

**Wulfram Gerstner**
School of Computer and Communication Sciences, Brain Mind Institute
École Polytechnique Fédérale de Lausanne
1015 Lausanne EPFL, Switzerland
`wulfram.gerstner@epfl.ch`

## Abstract

We derive a plausible learning rule for feedforward, feedback and lateral connections in a recurrent network of spiking neurons. Operating in the context of a generative model for distributions of spike sequences, the learning mechanism is derived from variational inference principles. The synaptic plasticity rules found are interesting in that they are strongly reminiscent of experimental Spike Time Dependent Plasticity, and in that they differ for excitatory and inhibitory neurons. A simulation confirms the method's applicability to learning both stationary and temporal spike patterns.

## 1 Introduction

This study considers whether recurrent networks of spiking neurons can be seen as a generative model not only of stationary patterns but also of temporal sequences. More precisely, we derive a model that learns to adapt its spontaneously spike sequences to conform as closely as possible to the empirical distribution of actual spike sequences caused by inputs impinging upon the sensory layer of the network.

A generative model is a model of the joint distribution of percepts and hidden causes in the world. Since the world has complex temporal relationships, we need a model that is able to both recognize and predict temporal patterns. Behavioural studies (e.g., [1]) support the assumption that the brain is performing approximate Bayesian inference. More recently, evidence for this hypothesis was found in electro-physiological work as well [2]. Various abstract Bayesian models have been proposed to account for this phenomenon [3, 4, 5, 6, 7]. However, it remains an open question whether optimization in abstract Bayesian models can be translated into plausible learning rules for synapses in networks of spiking neurons.

In this paper, we show that the derivation of spike-based plasticity rules from statistical learning principles yields learning dynamics for a generative spiking network model which are akin to those

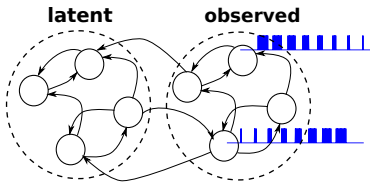

Figure 1: A network of spiking neurons, divided into observed and latent pools of neurons.

of Spike-Time Dependent Plasticity (STDP) [8]. Our learning rule is derived from a variational optimization process. Typically, optimization in recurrent Bayesian networks involves both forward and backward propagation steps. We propose a plasticity rule that approximates backward steps by the introduction of delayed updates in the synaptic weights and dynamics. The theory is supported by simulations in which we demonstrate that the learning mechanism is able to capture the hidden causes behind the observed spiking patterns.

We use the *Spike Response Model* (SRM) [9, 10], in which spikes are generated stochastically depending on the neuronal membrane potential. The SRM is an example of a generalized linear model (GLM). It is closely related to the integrate-and-fire model, and has been successfully used to explain neuronal spike trains [11, 12]. In this model, the membrane potential of a neuron $i$ at time $t$, expressed as $u_i(t)$ is given by

$$\tau \dot{u}_i(t) = -u_i(t) + b_i + \sum_j W_{i,j} X_j(t), \tag{1}$$

where $b_i$ is a bias which represents a constant external input to the neuron, and $X_j(t)$ is the spike train of the $j$th neuron defined by $X_j(t) = \sum_{t_j^f \in \{t_j^1, \ldots, t_j^N\}} \delta(t - t_j^f)$, where $\{t_j^1, \ldots, t_j^N\}$ is the set of spike timings. The diagonal elements of the synaptic matrix are kept fixed to a negative value $W_{i,i} = -\eta_0$ with $\eta_0 = 1.0$, which implements a reset of the membrane potential after each spike and is a simple way to take into account neuronal refractoriness [9, 13]. The time constant is taken to be $\tau = 10ms$ as in [13]. The spike generation process is stochastic with time-dependent firing intensity $\rho_i(t)$ which depends on the membrane potential $u_i(t)$:

$$\rho_i(t) = \rho_0 \exp(u_i(t)). \tag{2}$$

An exponential dependence of the firing intensity upon the membrane potential agrees with experimental results [12]. The set of equations (2) and (1) captures the simplified dynamics of a spiking neuron with stochastic spike timing.

In the following sections, we will introduce the theoretical framework and the approximations used in this paper. The basic learning mechanism is introduced and derived, followed by a simulation illustrating that our proposed learning rule is able to learn spatio-temporal features in the input spike trains and reproduce them in its spontaneous activity.

## 2 Principled Framework

We consider a network consisting of two distinct sets of neurons, *observed* neurons ( also called visible neurons or $\mathcal{V}$) and *latent* neurons ( also called hidden or $\mathcal{H}$), as illustrated in Figure 1. The activities of the observed neurons represent the quantity of interest to be modelled, while the latent neurons fulfill a mediating role representing the hidden causes of the observed spike train.

Learning in the context of this neuronal network consists of changing the synaptic strengths between neurons. We postulate that the underlying principle behind learning relies on learning *distributions* of spike trains evoked by either sensory inputs or more complicated sequences of cognitive events. In statistics, learning distributions involves minimizing a measure of distance between the model (that is, our neuronal network) and a target distribution (e.g. observations). A principled measure of distance between two distributions $p$ and $p_{\text{empirical}}$ is the Kullback-Leibler divergence [14] defined as

$$KL(p_{\text{empirical}}||p) = \int \mathcal{D}X p_{\text{empirical}}(X) \log \frac{p_{\text{empirical}}(X)}{p(X)}. \tag{3}$$

where individual $X$ represent entire spike trains. $\mathcal{D}X$ is a measure of integration over spike trains.

Our learning mechanism tries to minimize the KL divergence between the distribution defined by our network $p(X)$ and the observed spike timings distribution $p_{\text{empirical}}$ that is evoked by an unknown external process. Note that minimizing the KL divergence entails maximizing the likelihood that the observed spike trains $X_\mathcal{V}$ could have been generated by the model.

In order to derive the learning dynamics of our model in the next section, we need to evaluate the gradient of the likelihood (3) with respect to the free parameters of our model, i.e. the synaptic efficacies $W_{i,j}$ and biases $b_i$.

The joint likelihood of a particular spike train of both the observed $X_\mathcal{V}$ and the latent neurons $X_\mathcal{H}$ under our neuronal model can be written as [13]

$$\log p(X_\mathcal{V}, X_\mathcal{H}) = \sum_{i \in \mathcal{V} \cup \mathcal{H}} \int_0^T d\tau \left[ \log \rho_i(\tau) X_i(\tau) - \rho_i(\tau) \right] \tag{4}$$

Since we have a neuronal network including latent units (that is, neurons not receiving external inputs), the actual observation likelihood is an *effective* quantity obtained by integrating over all possible latent spike trains $X_\mathcal{H}$,

$$p(X_\mathcal{V}) = \int \mathcal{D}X_\mathcal{H} p(X_\mathcal{V}, X_\mathcal{H}). \tag{5}$$

The gradient of (5) is given by an expectation conditioned on the observed neurons' history:

$$\nabla \log p(X_\mathcal{V}) \quad = \quad \nabla \log \int \mathcal{D}X_\mathcal{H} p(X) \quad = \quad \langle \nabla \log p(X) \rangle_{p(X_\mathcal{H}|X_\mathcal{V})}$$

where $\langle f(X) \rangle_p = \int \mathcal{D}X f(x) p(x)$. This is difficult to evaluate since it conditions an entire latent spike train on an entire observed spike train. In other words, the posterior distribution of spike-timings of the latent neurons depends on both *past and future* of the observed neurons' spike train.

## 2.1  Weak Coupling Approximation

In order to render the model more tractable, we introduce an approximation on the dynamics based on the weak coupling approximation [15], which amounts to replacing (1) by

$$\tau \dot{u}_i(t) \quad = \quad -u_i(t) + b_i + \sum_j W_{i,j} \rho_j(t) + z_j(t), \tag{6}$$

where $z_i(t)$ is a Gaussian process with mean zero and inverse variance [1] $\lambda_i(t)$ given by

$$\lambda_i^{-1}(t) = \sigma_0 + \frac{1}{\tau^2} \sum_j W_{i,j}^2 \rho_j(t), \tag{7}$$

where $\sigma_0$ is intrinsic noise which we have added to regularize the simulations (we assume $\sigma_0 = 0.1$). Note that $\lambda_i(t)$ is a function of both the network state and synaptic efficacies. Our network model defines a joint distribution between observed input spike trains and membrane potentials given by

$$\log p(X_\mathcal{V}, u) = \sum_{i \in \mathcal{V}} \int dt \left[ X_i(t) u_i(t) - \rho_0 \exp(u_i(t)) \right] - \sum_{i \in \mathcal{V} \cup \mathcal{H}} \int dt \frac{\lambda_i(t)}{2} (\dot{u}_i(t) - f_i(t))^2, \tag{8}$$

where terms not depending on the model parameters and latent states have been dropped out as they do not contribute to the gradients we are interested in and $f_i(t)$ is the drift of the Gaussian process of the membrane potentials and can be read from equation (6). It is given by

$$f_i(t) = \frac{1}{\tau} \left( -u_i(t) + b_i + \sum_j W_{i,j} \rho_j(t) \right) \tag{9}$$

$$Var(u(t+dt)|u(t)) = \sum_j W_{i,j}^2 \int_t^{t+dt} ds \exp(2(s - t - dt)/\tau)(\rho_j(t))/\tau^2 = \frac{dt}{\tau^2} \sum_j W_{i,j}^2 \rho_j(t)$$

The weak coupling approximation amounts to replacing spikes of the latent neurons by intensities plus Gaussian noise. Note that in this approximated model, the latent variables are non longer the latent spike trains, but the membrane potentials. However, we emphasize that in the end the intensities can be substituted by spikes as we will see below.

## 2.2 Variational Approximation of the Posterior Membrane Potential $p(u|X_\mathcal{V})$

The variational approach in statistics is a method to approximate some complex distribution $p$ by a family of simpler distributions $q$. Variational methods have been applied to spiking neural networks in many different contexts, such as in connectivity or external source inference [20, 21]. In the following, we try to interpret the neural activity and plasticity together as an approximate form of variational learning.

We approximate the posterior $p(u|X_\mathcal{V})$ by the Gaussian process

$$\log q(u) \quad = \quad \sum_i \int dt \frac{\lambda_i(t)}{2} \left( \dot{u}_i(t) - h_i(t) \right)^2 + c \tag{10}$$

where the $h_i(t)$ are variational parameters representing the drift of the $ith$ membrane potential at time $t$ in the posterior process and $c$ is a normalization constant. Note that the parameters $\lambda_i(t)$ of the posterior process are taken to be the same as the network dynamics noise in (6). This is necessary in order to have a finite KL-divergence between the prior and the posterior processes [22].

Finding a good approximation for the variational parameters $h_i(t)$ amounts to minimizing the quantity $KL(q(u) \parallel p(X_\mathcal{V}, u))$, which is given by

$$\mathrm{KL}(q \parallel p) \quad = \quad \int dt \left\langle - \sum_{i \in \mathcal{V}} [X_i(t)u_i(t) - \rho_0 \exp(u_i(t))] \right.$$
$$\left. + \sum_{i \in \mathcal{V} \cup \mathcal{H}} \frac{\lambda_i(t)}{2} (\dot{u}_i(t) - f_i(t))^2 - \sum_{i \in \mathcal{V} \cup \mathcal{H}} \frac{\lambda_i(t)}{2} (\dot{u}_i(t) - h_i(t))^2 \right\rangle_{q(u)} \tag{11}$$

Although (11) can be written analytically in terms of the instantaneous mean and covariance of the posterior process, we adopt a simpler mean-field approximation, i.e. $\langle F(u_i(t)) \rangle \approx F(\langle u_i(t) \rangle)$. We write the mean $\langle u_i(t) \rangle = \bar{u}_i(t)$ as

$$\bar{u}_i(t) = \bar{u}_i(0) + \int_0^t ds\, h_i(s) \tag{12}$$

where the $h_i$ plays the role of the 'drift' or the derivative of $\bar{u}_i$. Note that $\frac{\delta \bar{u}_i(t)}{\delta h_j(t')} = \Theta(t - t')\delta_{i,j}$, where $\Theta(x)$ is the Heaviside step function. As a result, the KL-divergence becomes

$$\mathrm{KL}(q \parallel p) \quad \approx \quad \int dt \sum_i \left\{ -[X_i(t)\bar{u}_i(t) - \rho_0 \exp(\bar{u}_i(t))] \delta_{i \in \mathcal{V}} + \frac{\lambda_i(t)}{2} (h_i(t) - f_i(t))^2 \right\} \tag{13}$$

The drifts $h_i(t)$ of the variational approximation can be updated using gradient descent

$$\frac{\delta}{\delta h_k(t')} \mathrm{KL} \quad = \quad - \int dt \left[ X_k(t) - \rho_0 \exp(\bar{u}_k(t)) \right] \Theta(t - t')\delta_{k \in \mathcal{V}}$$
$$+ \lambda_k(t')(h_k(t') - f_k(t')) - \int dt \sum_i \lambda_i(t)(h_i(t) - f_i(t)) \frac{\delta f_i(t)}{\delta h_k(t')}$$
$$+ \frac{1}{2} \sum_i \int dt \frac{\delta \lambda_i(t)}{\delta h_k(t')} (h_i(t) - f_i(t))^2, \tag{14}$$

where

$$\frac{\delta f_i(t)}{\delta h_k(t')} \quad = \quad \frac{1}{\tau} \left( -\delta_{i,k} + W_{i,k}\rho_k(t) \right) \Theta(t - t') \tag{15}$$

$$\frac{\delta \lambda_i(t)}{\delta h_k(t')} \quad = \quad -\frac{1}{\tau^2} \lambda_i^2(t) W_{i,k}^2 \rho_k(t) \Theta(t - t') \tag{16}$$

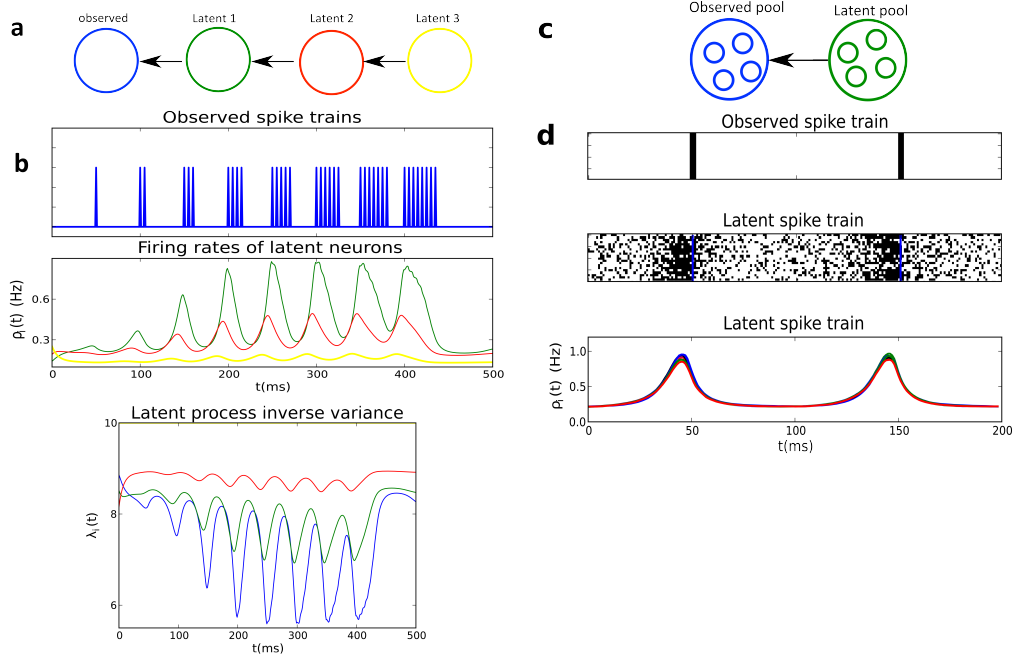

Figure 2: Posterior firing intensity for two simple networks: **(a)** A network with 4 neurons, simulated with mean field approximation. **(b)** From top to bottom: the observed spike train, the firing intensity for the three latent neurons and the posterior inverse variance. The green neuron has a direct connection to the observed neuron, and as such has a much stronger modulation of its firing rate than the other two latent neurons. **(c)** A network with two pools of 20 neurons, the observed and the latent pools. **(d)** Simulation results. From top to bottom: observed spike trains, spike trains in the latent pool and mean firing intensities of the latent neurons over different realizations of the network. The rate of the latent pool increases just before the spikes of the observed neurons. Note that the spiking implementation of the model has the same rates as the mathematical rate model.

There are few key points to note regarding (14). First, in the absence of observations, the best approximating $h_i(t)$ is simply given by $f_i(t)$, that is *the posterior and the prior processes become equal*. Second, the first, third and fourth terms in (14) are backward terms, that is, they correspond to corrections in the "belief" about the past states generated by new inputs. This implies that in order to estimate the drift $h_i(t)$ of the posterior membrane potential of neuron $i$ at time $t$, we need to know the observations $X(t')$ at time $t' > t$. Third, the fourth term in equation (14) is a contribution to the gradient that comes from the fact that the inverse variance $\lambda_i(t)$ defined in equation (7) is also a function of the network state. This is an important feature of the model, since it implies that the amount of noise in the dynamics is also being adapted to better explain the observed spike trains.

## 2.3 Towards Spike-time Dependent Plasticity

We learn the parameters of our network, that is, the synaptic weights and the neural 'biases' by gradient descent with learning rate $\eta$:

$$\Delta b_i = -\eta \frac{\delta}{\delta b_i} \text{KL} = -\eta \int dt \frac{\lambda_i(t)}{\tau}(h_i(t) - f_i(t)) \tag{17}$$

$$\Delta W_{k,l} = -\eta \frac{\delta}{\delta W_{k,l}} \text{KL} = -\eta \int dt \frac{\lambda_k(t)}{\tau}(h_k(t) - f_k(t))\rho_l(t)$$

$$+\eta \frac{1}{2} \sum_i \int dt \frac{\delta \lambda_i(t)}{\delta W_{k,l}}(h_i(t) - f_i(t))^2, \tag{18}$$

where $\frac{\delta \lambda_i(t)}{\delta W_{k,l}} = -2\frac{1}{\tau^2}\lambda_i^2(t)W_{i,l}\rho_l(t)\delta_{k,i}$. Note that once the posterior drift $h_i(t)$ is known, the computation of $\Delta b$ and $\Delta W$ can be done purely locally.

A long 'backward window' would, of course, be biologically implausible. However, on-line approximations to the backward terms provide a reasonable approximation by taking small backwards filters of up to $50ms$. Mechanistically, applications of $\Delta W$ can operate with a small delay, which is required to calculate the backwards correction term. In biology such delays indeed exist, as the weights are switched to a new value only some time *after* the stimulation that induces the change [23, 24]

More precisely, using a small backward window amounts to approximating the gradient of the posterior drift $h_i(t)$ by cutting off the time integrals using a finite time horizon, i.e., in equation (14) we replace integral $\int dt$ by $\int_{t'}^{t'+\Delta T} dt$ where $\Delta T$ is the size of the "backward window" used to approximate the gradient. The expression (14) can now be written as a delayed update equation

$$
\begin{aligned}
\delta h_k(t - \Delta T) \quad \propto \quad & -\int_{t-\Delta T}^{t} ds\, [X_k(s) - \rho_0 \exp(\bar{u}_k(s))]\, \delta_{k \in \mathcal{V}} \\
& + \lambda_k(t - \Delta T)(h_k(t - \Delta T) - f_k(t - \Delta T)) \\
& - \int_{t-\Delta T}^{t} ds \sum_i \lambda_i(s)(h_i(s) - f_i(s)) \frac{\delta f_i(s)}{\delta h_k(t - \Delta T)} \\
& + \frac{1}{2} \sum_i \int_{t-\Delta T}^{t} ds \frac{\delta \lambda_i(s)}{\delta h_k(t - \Delta T)} (h_i(s) - f_i(s))^2,
\end{aligned}
\qquad (19)
$$

The resulting update for the variable $h_k$ is used in the learning equation 18.

The simulation shown in Figure 2 provides a conceptual illustration of how the posterior firing intensity $\rho_l(t)$ propagates information backward from observed into latent neurons, a process that is essential for learning temporal patterns. Note that $\rho_l$ is the firing rate of the presynaptic neuron $l$ and as such it is information that is not directly available at the site of the synapse which has only access to spike arrivals (but not the underlying firing rate). However, spike arrivals do provide a reasonable estimate of the rate. Indeed Figure 2c and d show that a simulation of a network of pools of spiking neurons where updates are only based on spike times (rather than rates) gives qualitatively the same information as the rate formula derived above. In equations (20,15) we could therefore replace the pre-synaptic firing intensity $\rho_j(t)$ by temporally filtered spike trains which constitute a good approximation to $\rho_j(t)$.

## 2.4 STDP Window

From our learning equation for the synaptic weight (18), we can extract an STDP-like learning window by rewriting the plasticity rules as $\Delta W_{i,j} = \int dt \Delta W_{i,j}(t)$, where

$$
\Delta W_{i,j}(t) = \frac{\lambda_i(t)}{\tau}(h_i(t) - f_i(t))\rho_j(t) + \frac{1}{2} \sum_k \frac{\delta \lambda_k(t)}{\delta W_{i,j}}(h_k(t) - f_k(t))^2
\qquad (20)
$$

$\Delta W_{i,j}(t)$ is the expected change in $\Delta W_{i,j}$ at time $t$ under the posterior. As before, we replace the firing intensity $\rho_j$ in a given trial by the spikes. Assuming a spike of the observed neuron at $t = 0$, we have evaluated $h(t)$ and $f(t)$ and plot the weight change $\lambda_k(t')(h_k(t') - f_k(t'))$ that would occur if the latent neuron fires at $t'$ cf. equation (18). We show the resulting Spike-time Dependent Plasticity for a simple network of two neurons in Figure 3.

Note that the shape of $\Delta W_{i,j}(t)$ is remarkably reminiscent of the experimentally found measurements for STDP [8]. In particular, the shape of the STDP curve depends on the type of neuron and is different for connections from excitatory to excitatory than from excitatory to inhibitory or inhibitory to inhibitory neurons (Figure 3).

# 3 Simulations

In order to demonstrate the method's ability to capture both stationary and temporal patterns, we performed simulations on two tasks. The first one involves the formation of a temporal chain, while the second one involves a stationary pattern generator. Both simulations were done using a discrete-time (Euler method) version of the equations (14, 17, 18 and 19) with $dt = 1ms$. The backward window size was taken to be $\Delta T = 50ms$, and a learning rate of 0.02 was used.

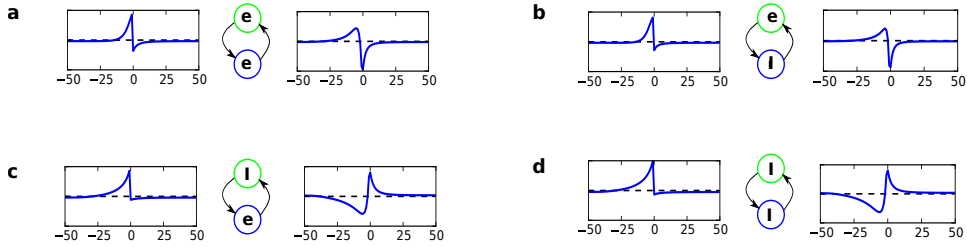

Figure 3: Spike-time Dependent Plasticity in a simple network composed of two neurons. Weight change $\Delta W_{i,j}(t)$ (vertical axis) as a function of spike timing of the neuron at the top (the latent neuron), given that the bottom (observed) neuron produces a spike at $t = 0$ (horizontal axis). Shown are all permutations of excitatory (e) and inhibitory (i) neuron types, with the left and right learning windows next to each network corresponding to the downward and upward synapses, respectively.

The first task consisted of learning a periodic chain, in which three pools of observed neurons were successively activated as shown in Figure 4a. A time lag was introduced between the third and the first pattern so as to force the network to form *temporal* hidden cause representations that are capable of capturing time dependencies without obvious observable instantaneous clues – during a blank moment, the only way a network can tell which pattern comes next is by actively using the latent neurons. After learning, the spontaneously patterns in the observable neurons developed a clear resemblance to the patterns provided during training, although a slightly larger amount of noise was present, as shown in Figure 4b. If the noise level of the model network is reduced, a noise-free "cleared-up concept" of the observed patterns is generated (Figure 4d) which clearly demonstrates that the recurrent network has indeed learned the task.

The way learning has configured the network in the sequence task can be understood if we study the connectivity pattern of the latent neurons. The latent neuron are active during the whole sequence (Figure 4c). We have reordered the labels of the neurons so that the structure of the connectivity matrix becomes as visble. There are subsets of latent neurons that are particularly active during each of the three 'subpatterns' in the sequence task, and other latent neurons that become active while the observable units are quiescent (Figure 4i). The lateral connectivity between the latent neurons has an asymmetry in the forward direction of the chain.

The second task aimed at learning to randomly generate one of three stationary patterns every $10ms$. Successfull learning of this task requires both the learning of the stationary patterns and the stochastic transitions between them. Figure 4d–g shows the results on this task.

# 4   Discussion

Some models have recently been proposed where STDP-like learning rules derive from 'first principles' (e.g., [25, 26, 13]). However, these models have either difficulty dealing with recurrent latent dynamics, or they do not account for non-factorial latent representations. In this work, we have proposed a plausible derivation for synaptic plasticity in a network consisting of spiking neurons, which can both capture time dependencies in observed spike trains and process combinatorial features. Using a generative model comprising both latent and observed neurons, the mechanism utilizes implicit (that is, short-term delayed) backward iterations that arise naturally from variational inference. A plasticity mechanism emerges that closely resembles that of the familiar STDP mechanism found in experimental studies. In our simulations we show that the plasticity rules are capable of learning both a temporal and a stationary pattern generator. Future work will attempt to further elucidate the possible biological plausibility of the approach, and its connection to Spike-Time Dependent Plasticity.

**Acknowledgments**

Support was provided by the SNF grant (CRSIK0 122697), the ERC grant (268689) and the SystemsX IPhD grant.

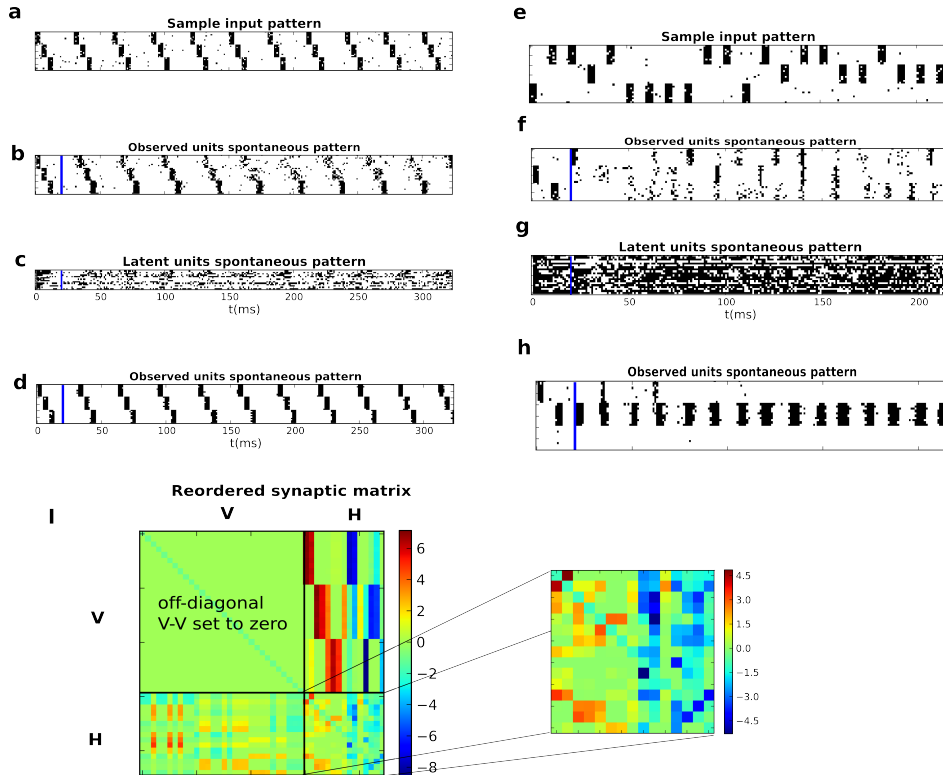

Figure 4: Simulation results. **Sequence task a–d, i**: a $20ms$-periodic sequence with a network of 30 observed neurons and 15 latent neurons having $50\%$ of inhibitory neurons (chosen randomly). The connections between the observed neurons have been set to zero in order to illustrate the use of latent-to-latent recurrent connections. **(a)** A sample of the periodic input pattern. Note the long waiting time after each sequence $1 - 2 - 3$ ($1 - 2 - 3 - \text{wait} - 1 - 2 - 3 - \dots$ ). **(b)** Simulations from the network with the first $20ms$ clamped to the data. **(c)** Latent neurons sample. **(d)** Sample simulation of the network with the *same parameters* but with less noise, in order to better show the underlying dynamics. This is achieved by the transformation $\rho_i(t) \rightarrow \rho_i(t)^\beta$ with $\beta = 2$. **Random jump task e–h**: learning to produce one of three patterns ($4ms$ long) every $10ms$. **(e)** A sample input pattern **(f)** One realization from the network with first the $20ms$ clamped to the data. **(g)** Sample latent pattern. **(h)** Sample simulation of the network with the *same parameters* but with less noise. Note that decreasing the level of noise is actually an impairment in performance for this task. **(i)** The learned synaptic matrix for the first task; the latent neurons have been re-ordered in order show the role of the latent-to-latent synapses in the dynamics as well as the role of the latent-to-observed synapses which represent the pattern features.

## Footnotes

[1] The variance of $\dot{u}$ due to the external input can be obtained by noting that $u_i(t+dt) = u_i(t) \exp(-dt/\tau) + \int_t^{t+dt} ds \exp((s - t - dt)/\tau)(b_i + \sum_j W_{i,j} X_j(t))/\tau$. Thus, in the weak coupling regime

# References

[1] Konrad P Körding and Daniel M Wolpert. Bayesian integration in sensorimotor learning. *Nature*, 427(6971):244–7, January 2004.

[2] P. Berkes, G. Orban, M. Lengyel, and J. Fiser. Spontaneous Cortical Activity Reveals Hallmarks of an Optimal Internal Model of the Environment. *Science*, 331(6013):83–87, January 2011.

[3] Wei Ji Ma, Jeffrey M Beck, and Alexandre Pouget. Spiking networks for Bayesian inference and choice. *Current opinion in neurobiology*, 18(2):217–22, April 2008.

[4] Joshua B Tenenbaum, Thomas L Griffiths, and Charles Kemp. Theory-based Bayesian models of inductive learning and reasoning. *Trends in cognitive sciences*, 10(7):309–18, 2006.

[5] Konrad P Körding and Daniel M Wolpert. Bayesian decision theory in sensorimotor control. *Trends in cognitive sciences*, 10(7):319–26, July 2006.

[6] D. Acuna and P. Schrater. Bayesian modeling of human sequential decision-making on the multi-armed bandit problem. In *Proceedings of the 30th Annual Conference of the Cognitive Science Society*. Washington, DC: Cognitive Science Society, 2008.

[7] Michael D. Lee. A Hierarchical Bayesian Model of Human Decision-Making on an Optimal Stopping Problem. *Cognitive Science*, 30(3):1–26, May 2006.

[8] G. Bi and M. Poo. Synaptic modification by correlated activity: Hebb's postulate revisited. *Annual review of neuroscience*, 24(1):139–166, 2001.

[9] W. Gerstner and W. K. Kistler. Mathematical Formulations of Hebbian Learning. *Biological Cybernetics*, 87(5-6):404–415, 2002. article.

[10] W. Gerstner. Spike-response model. *Scholarpedia*, 3(12):1343, 2008.

[11] J W Pillow, J Shlens, L Paninski, A Sher, A M Litke, E J Chichilnisky, and E P Simoncelli. Spatio-temporal correlations and visual signaling in a complete neuronal population. *Nature*, 454(7206):995–999, Aug 2008.

[12] Renaud Jolivet, Alexander Rauch, Hans R. Lüscher, and Wulfram Gerstner. Predicting spike timing of neocortical pyramidal neurons by simple threshold models. *J Comput Neurosci*, 21(1):35–49, August 2006.

[13] J.P. Pfister, Taro Toyoizumi, D. Barber, and W. Gerstner. Optimal spike-timing-dependent plasticity for precise action potential firing in supervised learning. *Neural Computation*, 18(6):1318–1348, 2006.

[14] S. Kullback and R. A. Leibler. On Information and Sufficiency. *The Annals of Mathematical Statistics*, 22(1):79–86, March 1951.

[15] Taro Toyoizumi, Kamiar Rahnama Rad, and Liam Paninski. Mean-field approximations for coupled populations of generalized linear model spiking neurons with Markov refractoriness. *Neural computation*, 21(5):1203–43, May 2009.

[16] Brendan J. Frey and Geoffrey E. Hinton. Variational Learning in Nonlinear Gaussian Belief Networks. *Neural Computation*, 11(1):193–213, January 1999.

[17] Karl Friston, Jérémie Mattout, Nelson Trujillo-Barreto, John Ashburner, and Will Penny. Variational free energy and the Laplace approximation. *NeuroImage*, 34(1):220–34, January 2007.

[18] Matthew J Beal and Zoubin Ghahramani. Variational Bayesian Learning of Directed Graphical Models with Hidden Variables. *Bayesian Analysis*, 1(4):793–832, 2006.

[19] T.S. Jaakkola and M.I. Jordan. Bayesian parameter estimation via variational methods. *Statistics and Computing*, 10(1):25–37, 2000.

[20] Jayant E Kulkarni and Liam Paninski. Common-input models for multiple neural spike-train data. *Network (Bristol, England)*, 18(4):375–407, December 2007.

[21] Ian H Stevenson, James M Rebesco, Nicholas G Hatsopoulos, Zach Haga, Lee E Miller, and Konrad P Körding. Bayesian inference of functional connectivity and network structure from spikes. *IEEE transactions on neural systems and rehabilitation engineering : a publication of the IEEE Engineering in Medicine and Biology Society*, 17(3):203–13, June 2009.

[22] C. Archambeau, Dan Cornford, Manfred Opper, and J. Shawe-Taylor. Gaussian process approximations of stochastic differential equations. In *Journal of Machine Learning Research Workshop and Conference Proceedings*, volume 1, pages 1–16. Citeseer, 2007.

[23] Daniel H O'Connor, Gayle M Wittenberg, and Samuel S-H Wang. Graded bidirectional synaptic plasticity is composed of switch-like unitary events. *Proceedings of the National Academy of Sciences of the United States of America*, 102(27):9679–84, July 2005.

[24] C C Petersen, R C Malenka, R a Nicoll, and J J Hopfield. All-or-none potentiation at CA3-CA1 synapses. *Proceedings of the National Academy of Sciences of the United States of America*, 95(8):4732–7, April 1998.

[25] Rajesh P N Rao. Bayesian computation in recurrent neural circuits. *Neural computation*, 16(1):1–38, January 2004.

[26] Bernhard Nessler, Michael Pfeiffer, and Wolfgang Maass. STDP enables spiking neurons to detect hidden causes of their inputs. *Advances in Neural Information Processing Systems (NIPS09)*, pages 1357–1365, 2009.

